# Analysis of Temporal-Difference Learning with Function Approximation

**John N. Tsitsiklis and Benjamin Van Roy**
Laboratory for Information and Decision Systems
Massachusetts Institute of Technology
Cambridge, MA 02139
e-mail: jnt@mit.edu, bvr@mit.edu

## Abstract

We present new results about the temporal-difference learning algorithm, as applied to approximating the cost-to-go function of a Markov chain using linear function approximators. The algorithm we analyze performs on-line updating of a parameter vector during a single endless trajectory of an aperiodic irreducible finite state Markov chain. Results include convergence (with probability 1), a characterization of the limit of convergence, and a bound on the resulting approximation error. In addition to establishing new and stronger results than those previously available, our analysis is based on a new line of reasoning that provides new intuition about the dynamics of temporal-difference learning. Furthermore, we discuss the implications of two counter-examples with regards to the significance of on-line updating and linearly parameterized function approximators.

## 1 INTRODUCTION

The problem of predicting the expected long-term future cost (or reward) of a stochastic dynamic system manifests itself in both time-series prediction and control. An example in time-series prediction is that of estimating the net present value of a corporation, as a discounted sum of its future cash flows, based on the current state of its operations. In control, the ability to predict long-term future cost as a function of state enables the ranking of alternative states in order to guide decision-making. Indeed, such predictions constitute the *cost-to-go function* that is central to dynamic programming and optimal control (Bertsekas, 1995).

Temporal-difference learning, originally proposed by Sutton (1988), is a method for approximating long-term future cost as a function of current state. The algorithm

is recursive, efficient, and simple to implement. Linear combinations of fixed basis functions are used to approximate the mapping from state to future cost. The weights of the linear combination are updated upon each observation of a state transition and the associated cost. The objective is to improve approximations of long-term future cost as more and more state transitions are observed. The trajectory of states and costs can be generated either by a physical system or a simulated model. In either case, we view the system as a Markov chain. Adopting terminology from dynamic programming, we will refer to the function mapping states of the Markov chain to expected long-term cost as the cost-to-go function.

In this paper, we introduce a new line of analysis for temporal-difference learning. In addition to providing new intuition about the dynamics of the algorithm, this approach leads to a stronger convergence result than previously available, as well as an interpretation of the limit of convergence and bounds on the resulting approximation error, neither of which have been available in the past. Aside from the statement of results, we maintain the discussion at an informal level, and make no attempt to present a complete or rigorous proof. The formal and more general analysis based on our line of reasoning can found in (Tsitsiklis and Van Roy, 1996), which also discusses the relationship between our results and other work involving temporal-difference learning.

The convergence results assume the use of both on-line updating and linearly parameterized function approximators. To clarify the relevance of these requirements, we discuss the implications of two counter-examples that are presented in (Tsitsiklis and Van Roy, 1996). These counter-examples demonstrate that temporal-difference learning can diverge in the presence of either nonlinearly parameterized function approximators or arbitrary (instead of on-line) sampling distributions.

## 2  DEFINITION OF TD($\lambda$)

In this section, we define precisely the nature of temporal-difference learning, as applied to approximation of the cost-to-go function for an infinite-horizon discounted Markov chain. While the method as well as our subsequent results are applicable to Markov chains with fairly general state spaces, including continuous and unbounded spaces, we restrict our attention in this paper to the case where the state space is finite. Discounted Markov chains with more general state spaces are addressed in (Tsitsiklis and Van Roy, 1996). Application of this line of analysis to the context of undiscounted absorbing Markov chains can be found in (Bertsekas and Tsitsiklis, 1996) and has also been carried out by Gurvits (personal communication).

We consider an aperiodic irreducible Markov chain with a state space $S = \{1, \ldots, n\}$, a transition probability matrix $P$ whose $(i, j)$th entry is denoted by $p_{ij}$, transition costs $g(i, j)$ associated with each transition from a state $i$ to a state $j$, and a discount factor $\alpha \in (0, 1)$. The sequence of states visited by the Markov chain is denoted by $\{i_t \mid t = 0, 1, \ldots\}$. The cost-to-go function $J^* : S \mapsto \Re$ associated with this Markov chain is defined by

$$J^*(i) \triangleq E\left[\sum_{t=0}^{\infty} \alpha^t g(i_t, i_{t+1}) \mid i_0 = i\right].$$

Since the number of dimensions is finite, it is convenient to view $J^*$ as a vector instead of a function.

We consider approximations of $J^*$ using a function of the form

$$\tilde{J}(i, r) = (\Phi r)(i).$$

Here, $r = (r(1), \ldots, r(K))$ is a parameter vector and $\Phi$ is a $n \times K$. We denote the $i$th row of $\Phi$ as a (column) vector $\phi(i)$.

Suppose that we observe a sequence of states $i_t$ generated according to the transition probability matrix $P$ and that at time $t$ the parameter vector $r$ has been set to some value $r_t$. We define the temporal difference $d_t$ corresponding to the transition from $i_t$ to $i_{t+1}$ by

$$d_t = g(i_t, i_{t+1}) + \alpha \tilde{J}(i_{t+1}, r_t) - \tilde{J}(i_t, r_t).$$

We define a sequence of *eligibility vectors* $z_t$ (of dimension $K$) by

$$z_t = \sum_{k=0}^{t} (\alpha\lambda)^{t-k} \phi(i_k).$$

The TD($\lambda$) updates are then given by

$$r_{t+1} = r_t + \gamma_t d_t z_t,$$

where $r_0$ is initialized to some arbitrary vector, $\gamma_t$ is a sequence of scalar step sizes, and $\lambda$ is a parameter in $[0, 1]$. Since temporal-difference learning is actually a continuum of algorithms, parameterized by $\lambda$, it is often referred to as TD($\lambda$). Note that the eligibility vectors can be updated recursively according to $z_{t+1} = \alpha\lambda z_t + \phi(i_{t+1})$, initialized with $z_{-1} = 0$.

## 3 ANALYSIS OF TD($\lambda$)

Temporal-difference learning originated in the field of reinforcement learning. A view commonly adopted in the original setting is that the algorithm involves "looking back in time and correcting previous predictions." In this context, the eligibility vector keeps track of how the parameter vector should be adjusted in order to appropriately modify prior predictions when a temporal-difference is observed. Here, we take a different view which involves examining the "steady-state" behavior of the algorithm and arguing that this characterizes the long-term evolution of the parameter vector. In the remainder of this section, we introduce this view of TD($\lambda$) and provide an overview of the analysis that it leads to. Our goal in this section is to convey some intuition about how the algorithm works, and in this spirit, we maintain the discussion at an informal level, omitting technical assumptions and other details required to formally prove the statements we make. These technicalities are addressed in (Tsitsiklis and Van Roy, 1996), where formal proofs are presented.

We begin by introducing some notation that will make our discussion here more concise. Let $\pi(1), \ldots, \pi(n)$ denote the steady-state probabilities for the process $i_t$. We assume that $\pi(i) > 0$ for all $i \in S$. We define an $n \times n$ diagonal matrix $D$ with diagonal entries $\pi(1), \ldots, \pi(n)$. We define a weighted norm $\| \cdot \|_D$ by

$$\|J\|_D = \sqrt{\sum_{i \in S} \pi(i) J^2(i)}.$$

We define a "projection matrix" $\Pi$ by

$$\Pi J = \arg \min_{\bar{J} = \Phi r} \|J - \bar{J}\|_D.$$

It is easy to show that $\Pi = \Phi(\Phi' D \Phi)^{-1} \Phi' D$.

We define an operator $T^{(\lambda)} : \Re^n \mapsto \Re^n$, indexed by a parameter $\lambda \in [0, 1)$ by

$$(T^{(\lambda)} J)(i) = (1 - \lambda) \sum_{m=0}^{\infty} \lambda^m E\left[ \sum_{t=0}^{m} \alpha^t g(i_t, i_{t+1}) + \alpha^{m+1} J(i_{m+1}) \mid i_0 = i \right].$$

For $\lambda = 1$ we define $(T^{(1)}J)(i) = J^*(i)$, so that $\lim_{\lambda \uparrow 1}(T^{(\lambda)}J)(i) = (T^{(1)}J)(i)$. To interpret this operator in a meaningful manner, note that, for each $m$, the term

$$E\left[\sum_{t=0}^{m}\alpha^t g(i_t, i_{t+1}) + \alpha^{m+1}J(i_{m+1}) \mid i_0 = i\right]$$

is the expected cost to be incurred over $m$ transitions plus an approximation to the remaining cost to be incurred, based on $J$. This sum is sometimes called the "$m$–stage truncated cost-to-go." Intuitively, if $J$ is an approximation to the cost-to-go function, the $m$–stage truncated cost-to-go can be viewed as an improved approximation. Since $T^{(\lambda)}J$ is a weighted average over the $m$–stage truncated cost-to-go values, $T^{(\lambda)}J$ can also be viewed as an improved approximation to $J^*$. A property of $T^{(\lambda)}$ that is instrumental in our proof of convergence is that $T^{(\lambda)}$ is a contraction of the norm $\|\cdot\|_D$. It follows from this fact that the composition $\Pi T^{(\lambda)}$ is also a contraction with respect to the same norm, and has a fixed point of the form $\Phi r^*$ for some parameter vector $r^*$.

To clarify the fundamental structure of TD($\lambda$), we construct a process $X_t = (i_t, i_{t+1}, z_t)$. It is easy to see that $X_t$ is a Markov process. In particular, $z_{t+1}$ and $i_{t+1}$ are deterministic functions of $X_t$ and the distribution of $i_{t+2}$ only depends on $i_{t+1}$. Note that at each time $t$, the random vector $X_t$, together with the current parameter vector $r_t$, provides all necessary information for computing $r_{t+1}$. By defining a function $s$ with $s(r, X) = (g(i, j) + \alpha \tilde{J}(j, r) - \tilde{J}(i, r))z$, where $X = (i, j, z)$, we can rewrite the TD($\lambda$) algorithm as

$$r_{t+1} = r_t + \gamma_t s(r_t, X_t).$$

For any $r$, $s(r, X_t)$ has a "steady-state" expectation, which we denote by $E_0[s(r, X_t)]$. Intuitively, once $X_t$ reaches steady-state, the TD($\lambda$) algorithm, in an "average" sense, behaves like the following deterministic algorithm:

$$\bar{r}_{\tau+1} = \bar{r}_\tau + \gamma_\tau E_0[s(\bar{r}_\tau, X_t)].$$

Under some technical assumptions, a theorem from (Benveniste, et al., 1990) can be used to deduce convergence TD($\lambda$) from that of the deterministic counterpart. Our study centers on an analysis of this deterministic algorithm. A theorem from (Benveniste, et al, 1990) is used to formally deduce convergence of the stochastic algorithm.

It turns out that

$$E_0[s(r, X_t)] = \Phi'D\Big(T^{(\lambda)}(\Phi r) - \Phi r\Big).$$

Using the contraction property of $T^{(\lambda)}$,

$$\begin{aligned}
(r - r^*)'E_0[s(r, X_t)] &= (\Phi r - \Phi r^*)'D\Big(\Pi T^{(\lambda)}(\Phi r) - \Phi r^* + (\Phi r^* - \Phi r)\Big) \\
&\leq \|\Phi r - \Phi r^*\|_D \cdot \|\Pi T^{(\lambda)}(\Phi r) - \Phi r^*\|_D - \|\Phi r^* - \Phi r\|_D^2 \\
&\leq (\alpha - 1)\|\Phi r - \Phi r^*\|_D^2.
\end{aligned}$$

Since $\alpha < 1$, this inequality shows that the steady state expectation $E_0[s(r, X_t)]$ generally moves the parameter vector towards $r^*$, the fixed point of $\Pi T^{(\lambda)}$, where "closeness" is measured in terms of the norm $\|\cdot\|_D$. This provides the main line of reasoning behind the proof of convergence provided in (Tsitsiklis and Van Roy, 1996). Some illuminating interpretations of this deterministic algorithm, which are useful in developing an intuitive understanding of temporal difference learning, are also discussed in (Tsitsiklis and Van Roy, 1996).

# 4　CONVERGENCE RESULT

We now present our main result concerning temporal-difference learning. A formal proof is provided in (Tsitsiklis and Van Roy, 1996).

**Theorem 1** *Let the following conditions hold:*
*(a) The Markov chain $i_t$ has a unique invariant distribution $\pi$ that satisfies $\pi'P = \pi'$, with $\pi(i) > 0$ for all $i$.*
*(b) The matrix $\Phi$ has full column rank; that is, the "basis functions" $\{\phi_k \mid k = 1, \ldots, K\}$ are linearly independent.*
*(c) The step sizes $\gamma_t$ are positive, nonincreasing, and predetermined. Furthermore, they satisfy $\sum_{t=0}^{\infty} \gamma_t = \infty$, and $\sum_{t=0}^{\infty} \gamma_t^2 < \infty$.*
*We then have:*
*(a) For any $\lambda \in [0,1]$, the TD($\lambda$) algorithm, as defined in Section 2, converges with probability 1.*
*(b) The limit of convergence $r^*$ is the unique solution of the equation*

$$\Pi T^{(\lambda)}(\Phi r^*) = \Phi r^*.$$

*(c) Furthermore, $r^*$ satisfies*

$$\|\Phi r^* - J^*\|_D \leq \frac{1 - \lambda\alpha}{1 - \alpha}\|\Pi J^* - J^*\|_D.$$

Part (b) of the theorem leads to an interesting interpretation of the limit of convergence. In particular, if we apply the TD($\lambda$) operator to the final approximation $\Phi r^*$, and then project the resulting function back into the span of the basis functions, we get the same function $\Phi r^*$. Furthermore, since the composition $\Pi T^{(\lambda)}$ is a contraction, repeated application of this composition to any function would generate a sequence of functions converging to $\Phi r^*$.

Part (c) of the theorem establishes that a certain desirable property is satisfied by the limit of convergence. In particular, if there exists a vector $r$ such that $\Phi r = J^*$, then this vector will be the limit of convergence of TD($\lambda$), for any $\lambda \in [0,1]$. On the other hand, if no such parameter vector exists, the distance between the limit of convergence $\Phi r^*$ and $J^*$ is bounded by a multiple of the distance between the projection $\Pi J^*$ and $J^*$. This latter distance is amplified by a factor of $(1 - \lambda\alpha)/(1 - \alpha)$, which becomes larger as $\lambda$ becomes smaller.

# 5　COUNTER-EXAMPLES

Sutton (1995) has suggested that on-line updating and the use of linear function approximators are both important factors that make temporal-difference learning converge properly. These requirements also appear as assumptions in the convergence result of the previous section. To formalize the fact that these assumptions are relevant, two counter-examples were presented in (Tsitsiklis and Van Roy, 1996).

The first counter-example involves the use of a variant of TD(0) that does not sample states based on trajectories. Instead, the states $i_t$ are sampled independently from a distribution $q(\cdot)$ over $S$, and successor states $j_t$ are generated by sampling according to $\Pr[j_t = j|i_t] = p_{i_tj}$. Each iteration of the algorithm takes on the form

$$r_{t+1} = r_t + \gamma_t\phi(i_t)\big(g(i_t, j_t) + \alpha\phi'(j_t)r_t - \phi'(i_t)r_t\big).$$

We refer to this algorithm as $q$–sampled TD(0). Note that this algorithm is closely related to the original TD($\lambda$) algorithm as defined in Section 2. In particular, if $i_t$ is

generated by the Markov chain and $j_t = i_{t+1}$, we are back to the original algorithm. It is easy to show, using a subset of the arguments required to prove Theorem 1, that this algorithm converges when $q(i) = \pi(i)$ for all $i$, and the Assumptions of Theorem 1 are satisfied. However, results can be very different when $q(\cdot)$ is arbitrary. In particular, the counter-example presented in (Tsitsiklis an Van Roy, 1996) shows that for any sampling distribution $q(\cdot)$ that is different from $\pi(\cdot)$ there exists a Markov chain with steady-state probabilities $\pi(\cdot)$ and a linearly parameterized function approximator for which $q$-sampled TD(0) diverges. A counter-example with similar implications has also been presented by Baird (1995).

A generalization of temporal difference learning is commonly used in conjunction with nonlinear function approximators. This generalization involves replacing each vector $\phi(i_t)$ that is used to construct the eligibility vector with the vector of derivatives of $\tilde{J}(i_t, \cdot)$, evaluated at the current parameter vector $r_t$. A second counter-example in (Tsitsiklis and Van Roy, 1996), shows that there exists a Markov chain and a nonlinearly parameterized function approximator such that both the parameter vector and the approximated cost-to-go function diverge when such a variant of TD(0) is applied. This nonlinear function approximator is "regular" in the sense that it is infinitely differentiable with respect to the parameter vector. However, it is still somewhat contrived, and the question of whether such a counter-example exists in the context of more standard function approximators such as neural networks remains open.

## 6    CONCLUSION

Theorem 1 establishes convergence with probability 1, characterizes the limit of convergence, and provides error bounds, for temporal-difference learning. It is interesting to note that the margins allowed by the error bounds are inversely proportional to $\lambda$. Although this is only a bound, it strongly suggests that higher values of $\lambda$ are likely to produce more accurate approximations. This is consistent with the examples that have been constructed by Bertsekas (1994).

The sensitivity of the error bound to $\lambda$ raises the question of whether or not it ever makes sense to set $\lambda$ to values less than 1. Many reports of experimental results, dating back to Sutton (1988), suggest that setting $\lambda$ to values less than one can often lead to significant gains in the rate of convergence. A full understanding of how $\lambda$ influences the rate of convergence is yet to be found, though some insight in the case of look-up table representations is provided by Dayan and Singh (1996). This is an interesting direction for future research.

### Acknowledgments

We thank Rich Sutton for originally making us aware of the relevance of on-line state sampling, and also for pointing out a simplification in the expression for the error bound of Theorem 1. This research was supported by the NSF under grant DMI-9625489 and the ARO under grant DAAL-03-92-G-0115.

### References

Baird, L. C. (1995). "Residual Algorithms: Reinforcement Learning with Function Approximation," in Prieditis & Russell, eds. Machine Learning: Proceedings of the Twelfth International Conference, 9-12 July, Morgan Kaufman Publishers, San Francisco, CA.

Bertsekas, D. P. (1994) "A Counter-Example to Temporal-Difference Learning,"

Neural Computation, vol. 7, pp. 270-279.

Bertsekas, D. P. (1995) *Dynamic Programming and Optimal Control*, Athena Scientific, Belmont, MA.

Bertsekas, D. P. & Tsitsiklis, J. N. (1996) *Neuro-Dynamic Programming*, Athena Scientific, Belmont, MA.

Benveniste, A., Metivier, M., & Priouret, P., (1990) *Adaptive Algorithms and Stochastic Approximations*, Springer-Verlag, Berlin.

Dayan, P. D. & Singh, S. P (1996) "Mean Squared Error Curves in Temporal Difference Learning," preprint.

Gurvits, L. (1996) personal communication.

Sutton, R. S., (1988) "Learning to Predict by the Method of Temporal Differences," Machine Learning, vol. 3, pp. 9-44.

Sutton, R.S. (1995) "On the Virtues of Linear Learning and Trajectory Distributions," Proceedings of the Workshop on Value Function Approximation, Machine Learning Conference 1995, Boyan, Moore, and Sutton, Eds., p. 85. Technical Report CMU-CS-95-206, Carnegie Mellon University, Pittsburgh, PA 15213.

Tsitsiklis, J. N. & Van Roy, B. (1996) "An Analysis of Temporal-Difference Learning with Function Approximation," to appear in the *IEEE Transactions on Automatic Control*.
